# Discriminative Fields for Modeling Spatial Dependencies in Natural Images

**Sanjiv Kumar and Martial Hebert**
The Robotics Institute
Carnegie Mellon University
Pittsburgh, PA 15213
{skumar,hebert}@ri.cmu.edu

## Abstract

In this paper we present Discriminative Random Fields (DRF), a discriminative framework for the classification of natural image regions by incorporating neighborhood spatial dependencies in the labels as well as the observed data. The proposed model exploits local discriminative models and allows to relax the assumption of conditional independence of the observed data given the labels, commonly used in the Markov Random Field (MRF) framework. The parameters of the DRF model are learned using penalized maximum pseudo-likelihood method. Furthermore, the form of the DRF model allows the MAP inference for binary classification problems using the graph min-cut algorithms. The performance of the model was verified on the synthetic as well as the real-world images. The DRF model outperforms the MRF model in the experiments.

## 1 Introduction

For the analysis of natural images, it is important to use contextual information in the form of spatial dependencies in images. In a probabilistic framework, this leads one to random field modeling of the images. In this paper we address the main challenge involving such modeling, i.e. how to model arbitrarily complex dependencies in the observed image data as well as the labels in a principled manner.

In the literature, Markov Random Field (MRF) is a commonly used model to incorporate contextual information [1]. MRFs are generally used in a probabilistic generative framework that models the joint probability of the observed data and the corresponding labels. In other words, let $\boldsymbol{y}$ be the observed data from an input image, where $\boldsymbol{y} = \{\boldsymbol{y}_i\}_{i \in S}$, $\boldsymbol{y}_i$ is the data from $i^{th}$ site, and $S$ is the set of sites. Let the corresponding labels at the image sites be given by $\boldsymbol{x} = \{x_i\}_{i \in S}$. In the MRF framework, the posterior over the labels given the data is expressed using the Bayes' rule as, $p(\boldsymbol{x}|\boldsymbol{y}) \propto p(\boldsymbol{x}, \boldsymbol{y}) = p(\boldsymbol{x})p(\boldsymbol{y}|\boldsymbol{x})$ where the prior over labels, $p(\boldsymbol{x})$ is modeled as a MRF. For computational tractability, the observation or likelihood model, $p(\boldsymbol{y}|\boldsymbol{x})$ is usually assumed to have a factorized form, i.e. $p(\boldsymbol{y}|\boldsymbol{x}) = \prod_{i \in S} p(\boldsymbol{y}_i|x_i)$[1][2]. However, as noted by several researchers [3][4], this assumption is too restrictive for the analysis of natural images. For example, consider a class that contains man-made structures (e.g. buildings). The data belonging to such a class is highly dependent on its neighbors since the lines or edges at spatially adjoining sites follow

some underlying organization rules rather than being random (See Fig. 2). This is also true for a large number of texture classes that are made of structured patterns.

Some efforts have been made in the past to model the dependencies in the data [3][4], but they make factored approximations of the actual likelihood for tractability. In addition, simplistic forms of the factors preclude capturing stronger relationships in the observations in the form of arbitrarily complex features that might be desired to discriminate between different classes. Now considering a different point of view, for classification purposes, we are interested in estimating the posterior over labels given the observations, i.e., $p(\boldsymbol{x}|\boldsymbol{y})$. In a generative framework, one expends efforts to model the joint distribution $p(\boldsymbol{x}, \boldsymbol{y})$, which involves implicit modeling of the observations. In a discriminative framework, one models the distribution $p(\boldsymbol{x}|\boldsymbol{y})$ directly. As noted in [2], a potential advantage of using the discriminative approach is that the true underlying generative model may be quite complex even though the class posterior is simple. This means that the generative approach may spend a lot of resources on modeling the generative models which are not particularly relevant to the task of inferring the class labels. Moreover, learning the class density models may become even harder when the training data is limited [5].

In this work we present a Discriminative Random Field (DRF) model based on the concept of Conditional Random Field (CRF) proposed by Lafferty et al. [6] in the context of segmentation and labeling of 1-D text sequences. The CRFs directly model the posterior distribution $p(\boldsymbol{x}|\boldsymbol{y})$ as a Gibbs field. This approach allows one to capture arbitrary dependencies between the observations without resorting to any model approximations. Our model further enhances the CRFs by proposing the use of local discriminative models to capture the class associations at individual sites as well as the interactions in the neighboring sites on 2-D grid lattices. The proposed model uses local discriminative models to achieve the site classification while permitting interactions in both the observed data and the label field in a principled manner. The research presented in this paper alleviates several problems with the previous version of the DRFs described in [7].

## 2   Discriminative Random Field

We first restate in our notations the definition of the Conditional Random Fields as given by Lafferty et al. [6]. In this work we will be concerned with binary classification, i.e. $x_i \in \{-1, 1\}$. Let the observed data at site $i$, $\boldsymbol{y}_i \in \Re^c$.

**CRF Definition**: *Let $G = (S, E)$ be a graph such that $\boldsymbol{x}$ is indexed by the vertices of $G$. Then $(\boldsymbol{x}, \boldsymbol{y})$ is said to be a conditional random field if, when conditioned on $\boldsymbol{y}$, the random variables $x_i$ obey the Markov property with respect to the graph: $p(x_i|\boldsymbol{y}, \boldsymbol{x}_{S-\{i\}}) = p(x_i|\boldsymbol{y}, \boldsymbol{x}_{\mathcal{N}_i})$, where $S - \{i\}$ is the set of all nodes in $G$ except the node $i$, $\mathcal{N}_i$ is the set of neighbors of the node $i$ in $G$, and $\boldsymbol{x}_\Omega$ represents the set of labels at the nodes in set $\Omega$.*

Thus CRF is a random field globally conditioned on the observations $\boldsymbol{y}$. The condition of positivity requiring $p(\boldsymbol{x}|\boldsymbol{y}) > 0 \ \forall \ \boldsymbol{x}$ has been assumed implicitly. Now, using the Hammersley Clifford theorem [1] and assuming only up to pairwise clique potentials to be nonzero, the joint distribution over the labels $\boldsymbol{x}$ given the observations $\boldsymbol{y}$ can be written as,

$$p(\boldsymbol{x}|\boldsymbol{y}) = \frac{1}{Z} \exp \left( \sum_{i \in S} A_i(x_i, \boldsymbol{y}) + \sum_{i \in S} \sum_{j \in \mathcal{N}_i} I_{ij}(x_i, x_j, \boldsymbol{y}) \right) \tag{1}$$

where $Z$ is a normalizing constant known as the partition function, and $-A_i$ and $-I_{ij}$ are the unary and pairwise potentials respectively. With a slight abuse of notations, in the rest of the paper we will call $A_i$ as *association potential* and $I_{ij}$ as *interaction potential*. Note that both the terms explicitly depend on all the observations $\boldsymbol{y}$. In the DRFs, the association potential is seen as a local decision term which decides the association of a given site to a certain class ignoring its neighbors. The interaction potential is seen as a data dependent

smoothing function. For simplicity, in the rest of the paper we assume the random field given in (1) to be homogeneous and isotropic, i.e. the functional forms of $A_i$ and $I_{ij}$ are independent of the locations $i$ and $j$. Henceforth we will leave the subscripts and simply use the notations $A$ and $I$. Note that the assumption of isotropy can be easily relaxed at the cost of a few additional parameters.

## 2.1 Association potential

In the DRF framework, $A(x_i, \boldsymbol{y})$ is modeled using a local discriminative model that outputs the association of the site $i$ with class $x_i$. Generalized Linear Models (GLM) are used extensively in statistics to model the class posteriors given the observations [8]. For each site $i$, let $\boldsymbol{f}_i(\boldsymbol{y})$ be a function that maps the observations $\boldsymbol{y}$ on a feature vector such that $\boldsymbol{f}_i : \boldsymbol{y} \to \Re^l$. Using a logistic function as the *link*, the local class posterior can be modeled as,

$$P(x_i = 1 | \boldsymbol{y}) = \frac{1}{1 + e^{-(w_0 + \boldsymbol{w}_1^T \boldsymbol{f}_i(\boldsymbol{y}))}} = \sigma(w_0 + \boldsymbol{w}_1^T \boldsymbol{f}_i(\boldsymbol{y})) \tag{2}$$

where $\boldsymbol{w} = \{w_0, \boldsymbol{w}_1\}$ are the model parameters. To extend the logistic model to induce a nonlinear decision boundary in the feature space, a transformed feature vector at each site $i$ is defined as, $\boldsymbol{h}_i(\boldsymbol{y}) = [1, \phi_1(\boldsymbol{f}_i(\boldsymbol{y})), \dots, \phi_R(\boldsymbol{f}_i(\boldsymbol{y}))]^T$ where $\phi_k(.)$ are arbitrary nonlinear functions. The first element of the transformed vector is kept as 1 to accommodate the bias parameter $w_0$. Further, since $x_i \in \{-1, 1\}$, the probability in (2) can be compactly expressed as $P(x_i | \boldsymbol{y}) = \sigma(x_i \boldsymbol{w}^T \boldsymbol{h}_i(\boldsymbol{y}))$. Finally, the association potential is defined as,

$$A(x_i, \boldsymbol{y}) = \log(\sigma(x_i \boldsymbol{w}^T \boldsymbol{h}_i(\boldsymbol{y})) \tag{3}$$

This transformation makes sure that the DRF yields standard logistic classifier if the interaction potential in (1) is set to zero. Note that the transformed feature vector at *each* site $i$, i.e. $\boldsymbol{h}_i(\boldsymbol{y})$ is a function of whole set of observations $\boldsymbol{y}$. On the contrary, the assumption of conditional independence of the data in the MRF framework allows one to use the data only from a particular site, i.e. $\boldsymbol{y}_i$ to get the log-likelihood, which acts as the association potential.

As a related work, in the context of tree-structured belief networks, Feng et al. [2] used the scaled likelihoods to approximate the actual likelihoods at each site required by the generative formulation. These scaled likelihoods were obtained by scaling the local class posteriors learned using a neural network. On the contrary, in the DRF model, the local class posterior is an integral part of the full conditional model in (1). Also, unlike [2], the parameters of the association and interaction potential are learned simultaneously in the DRF framework.

## 2.2 Interaction potential

To model the interaction potential $I$, we first analyze the interaction potential commonly used in the MRF framework. Note that the MRF framework does not permit the use of data in the interaction potential. For a homogeneous and isotropic Ising model, the interaction potential is given as $I = \beta x_i x_j$, which penalizes every dissimilar pair of labels by the cost $\beta$ [1]. This form of interaction prefers piecewise constant smoothing without explicitly considering discontinuities in the data. In the DRF formulation, the interaction potential is a function of all the observations $\boldsymbol{y}$. We would like to have similar labels at a pair of sites for which the observed data supports such a hypothesis. In other words, we are interested in learning a pairwise discriminative model as the interaction potential.

For a pair of sites $(i, j)$, let $\boldsymbol{\mu}_{ij}(\boldsymbol{\psi}_i(\boldsymbol{y}), \boldsymbol{\psi}_j(\boldsymbol{y}))$ be a new feature vector such that $\boldsymbol{\mu}_{ij} : \Re^\gamma \times \Re^\gamma \to \Re^q$, where $\boldsymbol{\psi}_k : \boldsymbol{y} \to \Re^\gamma$. Denoting this feature vector as $\boldsymbol{\mu}_{ij}(\boldsymbol{y})$ for simplification, the interaction potential is modeled as,

$$I(x_i, x_j, \boldsymbol{y}) = x_i x_j \boldsymbol{v}^T \boldsymbol{\mu}_{ij}(\boldsymbol{y}) \tag{4}$$

where $\boldsymbol{v}$ are the model parameters. Note that the first component of $\boldsymbol{\mu}_{ij}(\boldsymbol{y})$ is fixed to be 1 to accommodate the bias parameter. This form of interaction potential is much simpler than the one proposed in [7], and makes the parameter learning a convex problem. There are two interesting properties of the interaction potential given in (4). First, if the association potential at each site and the interaction potentials of all the pairwise cliques except the pair $(i, j)$ are set to zero in (1), the DRF acts as a logistic classifier which yields the probability of the site pair to have the same labels given the observed data. Second, the proposed interaction potential is a generalization of the Ising model. The original Ising form is recovered if all the components of vector $\boldsymbol{v}$ other than the bias parameter are set to zero in (4). Thus, the form in (4) acts as a data-dependent discontinuity adaptive model that will moderate smoothing when the data from the two sites is 'different'. The data-dependent smoothing can especially be useful to absorb the errors in modeling the association potential. Anisotropy can be easily included in the DRF model by parametrizing the interaction potentials of different directional pairwise cliques with different sets of parameters $\boldsymbol{v}$.

## 3 Parameter learning and inference

Let $\theta$ be the set of DRF parameters where $\theta = \{\boldsymbol{w}, \boldsymbol{v}\}$. The form of the DRF model resembles the posterior of the MRF framework assuming conditionally independent data. However, in the MRF framework, the parameters of the class generative models, $p(\boldsymbol{y}_i|x_i)$ and the parameters of the prior random field on labels, $p(\boldsymbol{x})$ are generally assumed to be independent and learned separately [1]. In contrast, we make no such assumption and learn all the parameters of the DRF simultaneously.

The maximum likelihood approach to learn the DRF parameters involves evaluation of the partition function $Z$ which is in general a NP-hard problem. One could use either sampling techniques or resort to some approximations e.g. pseudo-likelihood to estimate the parameters. In this work we used the pseudo-likelihood formulation due to its simplicity and consistency of the estimates for the large lattice limit [1]. In the pseudo-likelihood approach, a factored approximation is used such that, $P(\boldsymbol{x}|\boldsymbol{y}, \theta) \approx \prod_{i \in S} P(x_i|\boldsymbol{x}_{\mathcal{N}_i}, \boldsymbol{y}, \theta)$. However, for the Ising model in MRFs, pseudo-likelihood tends to overestimate the interaction parameter $\beta$, causing the MAP estimates of the field to be very poor solutions [9]. Our experiments in the previous work [7] and Section 4 of this paper verify these observations for the interaction parameters in DRFs too. To alleviate this problem, we take a Bayesian approach to get the maximum a posteriori estimates of the parameters. Similar to the concept of weight decay in neural learning literature, we assume a Gaussian prior over the interaction parameters $\boldsymbol{v}$ such that $p(\boldsymbol{v}|\tau) = \mathcal{N}(\boldsymbol{v}; \boldsymbol{0}, \tau^2 \boldsymbol{I})$ where $\boldsymbol{I}$ is the identity matrix. Using a prior over parameters $\boldsymbol{w}$ that leads to weight decay or shrinkage might also be beneficial but we leave that for future exploration. The prior over parameters $\boldsymbol{w}$ is assumed to be uniform. Thus, given $M$ independent training images,

$$\widehat{\theta} = \arg\max_{\theta} \sum_{m=1}^{M} \sum_{i \in S} \left\{ \log \sigma(x_i \boldsymbol{w}^T \boldsymbol{h}_i(\boldsymbol{y})) + \sum_{j \in \mathcal{N}_i} x_i x_j \boldsymbol{v}^T \boldsymbol{\mu}_{ij}(\boldsymbol{y}) - \log z_i \right\} - \frac{1}{2\tau^2} \boldsymbol{v}^T \boldsymbol{v} \quad (5)$$

$$\text{where} \quad z_i = \sum_{x_i \in \{-1,1\}} \exp \left\{ \log \sigma(x_i \boldsymbol{w}^T \boldsymbol{h}_i(\boldsymbol{y})) + \sum_{j \in \mathcal{N}_i} x_i x_j \boldsymbol{v}^T \boldsymbol{\mu}_{ij}(\boldsymbol{y}) \right\}$$

If $\tau$ is given, the penalized log pseudo-likelihood in (5) is convex with respect to the model parameters and can be easily maximized using gradient descent.

As a related work regarding the estimation of $\tau$, Mackay [10] has suggested the use of type II marginal likelihood. But in the DRF formulation, integrating the parameters $\boldsymbol{v}$ is a hard problem. Another choice is to integrate out $\tau$ by choosing a non-informative

hyperprior on $\tau$ as in [11] [12]. However our experiments showed that these methods do not yield good estimates of the parameters because of the use of pseudo-likelihood in our framework. In the present work we choose $\tau$ by cross-validation. Alternative ways of parameter estimation include the use of contrastive divergence [13] and saddle point approximations resembling perceptron learning rules [14]. We are currently exploring these possibilities.

The problem of inference is to find the optimal label configuration $\boldsymbol{x}$ given an image $\boldsymbol{y}$, where optimality is defined with respect to a cost function. In the current work we use the MAP estimate as the solution to the inference problem. While using the Ising MRF model for the binary classification problems, exact MAP solution can be computed using min-cut/max-flow algorithms provided $\beta \geq 0$ [9][15]. For the DRF model, the MAP estimates can be obtained using the same algorithms. However, since these algorithms do not allow negative interaction between the sites, the data-dependent smoothing for each clique is set to be $\boldsymbol{v}^T \boldsymbol{\mu}_{ij}(\boldsymbol{y}) = \max\{0, \boldsymbol{v}^T \boldsymbol{\mu}_{ij}(\boldsymbol{y})\}$, yielding an approximate MAP estimate. This is equivalent to switching the smoothing off at the image discontinuities.

## 4 Experiments and discussion

For comparison, a MRF framework was also learned assuming a conditionally independent likelihood and a homogeneous and isotropic Ising interaction model. So, the MRF posterior is $p(\boldsymbol{x}|\boldsymbol{y}) = Z_m^{-1} \exp\left(\sum_{i \in S} \log p(\boldsymbol{s}_i(\boldsymbol{y}_i)|x_i) + \sum_{i \in S} \sum_{j \in \mathcal{N}_i} \beta x_i x_j\right)$ where $\beta$ is the interaction parameter and $\boldsymbol{s}_i(\boldsymbol{y}_i)$ is a *single-site* feature vector at $i^{th}$ site such that $\boldsymbol{s}_i : \boldsymbol{y}_i \rightarrow \Re^d$. Note that $\boldsymbol{s}_i(\boldsymbol{y}_i)$ does not take into account influence of the data in the neighborhood of $i^{th}$ site. A first order neighborhood (4 nearest neighbors) was used for label interaction in all the experiments.

### 4.1 Synthetic images

The aim of these experiments was to obtain correct labels from corrupted binary images. Four base images, $64 \times 64$ pixels each, were used in the experiments (top row in Fig. 1). We compare the DRF and the MRF results for two different noise models. For each noise model, $50$ images were generated from each base image. Each pixel was considered as an image site and the feature vector $\boldsymbol{s}_i(\boldsymbol{y}_i)$ was simply chosen to be a scalar representing the intensity at $i^{th}$ site. In experiments with the synthetic data, no neighborhood data interaction was used for the DRFs (i.e. $\boldsymbol{f}_i(\boldsymbol{y}) = \boldsymbol{s}_i(\boldsymbol{y}_i)$) to observe the gains only due to the use of discriminative models in the association and interaction potentials. A linear discriminant was implemented in the association potential such that $\boldsymbol{h}_i(\boldsymbol{y}) = [1, \boldsymbol{f}_i(\boldsymbol{y})]^T$. The pairwise data vector $\boldsymbol{\mu}_{ij}(\boldsymbol{y})$ was obtained by taking the absolute difference of $\boldsymbol{s}_i(\boldsymbol{y}_i)$ and $\boldsymbol{s}_j(\boldsymbol{y}_j)$. For the MRF model, each class-conditional density, $p(\boldsymbol{s}_i(\boldsymbol{y}_i)|x_i)$, was modeled as a Gaussian. The noisy data from the left most base image in Fig.1 was used for training while $150$ noisy images from the rest of the three base images were used for testing.

Three experiments were conducted for each noise model. (i) The interaction parameters for the DRF ($\boldsymbol{v}$) as well as for the MRF ($\beta$) were set to zero. This reduces the DRF model to a logistic classifier and MRF to a maximum likelihood (ML) classifier. (ii) The parameters of the DRF, i.e. $[\boldsymbol{w}, \boldsymbol{v}]$, and the MRF, i.e. $\beta$, were learned using pseudo-likelihood approach without any penalty, i.e. $\tau = \infty$. (iii) Finally, the DRF parameters were learned using penalized pseudo-likelihood and the best $\beta$ for the MRF was chosen from cross-validation. The MAP estimates of the labels were obtained using graph-cuts for both the models.

Under the first noise model, each image pixel was corrupted with independent Gaussian noise of standard deviation $0.3$. For the DRF parameter learning, $\tau$ was chosen to be $0.01$. The pixelwise classification error for this noise model is given in the top row of Table 1. Since the form of noise is the same as the likelihood model in the MRF, MRF is

Table 1: Pixelwise classification errors (%) on 150 synthetic test images. For the Gaussian noise MRF and DRF give similar error while for 'bimodal' noise, DRF performs better. Note that only label interaction (i.e. no data interaction) was used for these tests (see text).

| Noise | ML | Logistic | MRF (PL) | DRF (PL) | MRF | DRF |
|---|---|---|---|---|---|---|
| Gaussian | 15.62 | 15.78 | 13.18 | 29.49 | 2.35 | 2.30 |
| Bimodal | 24.00 | 29.86 | 22.70 | 29.49 | 7.00 | 6.21 |

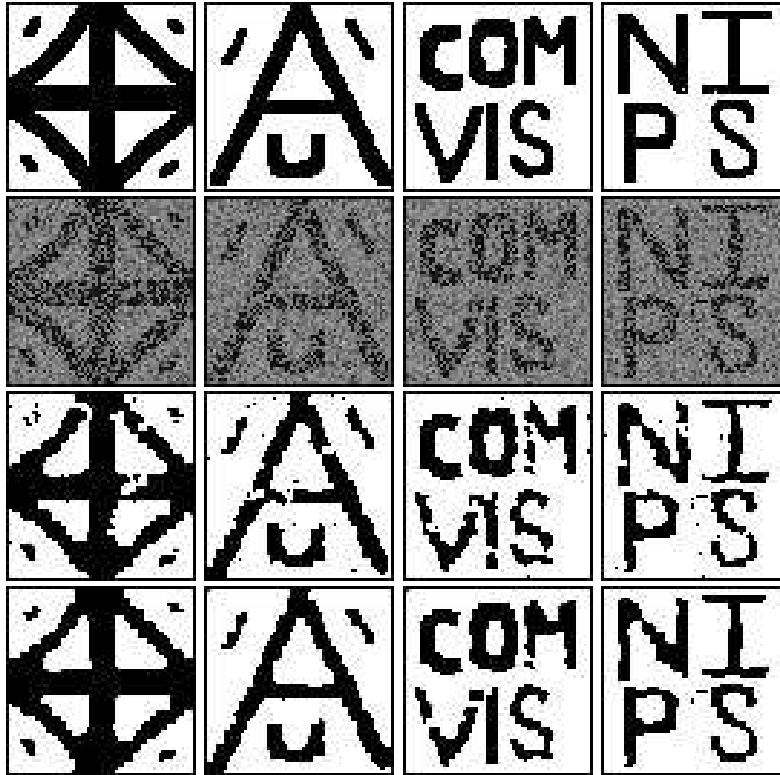

Figure 1: Results on synthetic images. From top, first row:original images, second row: images corrupted with 'bimodal' noise, third row: MRF results, fourth row: DRF results.

expected to give good results. The DRF model does marginally better than MRF even for this case. Note that the DRF and MRF results are worse when the parameters were learned without penalizing the pseudo-likelihood (shown in Table 1 with suffix (PL)). The MAP inference yields oversmoothed images for these parameters. The DRF model is affected more because all the parameters in DRFs are learned simultaneously unlike MRFs.

In the second noise model each pixel was corrupted with independent mixture of Gaussian noise. For each class, a mixture of two Gaussians with equal mixing weights was used yielding a 'bimodal' class noise. The mixture model parameters (mean, std) for the two classes were chosen to be $[(0.08, 0.03), (0.46, 0.03)]$, and $[(0.55, 0.02), (0.42, 0.10)]$ inspired by [5]. The classification results are shown in the bottom row of Table 1. An interesting point to note is that DRF yields lower error than MRF even when the logistic classifier has higher error than the ML classifier on the test data. For a typical noisy version of the four base images, the performance of different techniques in compared in Fig. 1.

Table 2: Detection Rates (DR) and False Positives (FP) for the test set containing 129 images $(49, 536$ sites). FP for logistic classifier were kept to be the same as for DRF for DR comparison. Superscript $'-'$ indicates no neighborhood data interaction was used.

|  | MRF | Logistic$^-$ | DRF$^-$ | Logistic | DRF |
|---|---|---|---|---|---|
| DR (%) | 58.35 | 47.50 | 61.79 | 60.80 | 72.54 |
| FP (per image) | 2.44 | 2.28 | 2.28 | 1.76 | 1.76 |

## 4.2 Real-World images

The proposed DRF model was applied to the task of detecting man-made structures in natural scenes. The aim was to label each image site as *structured* or *nonstructured*. The training and the test set contained 108 and 129 images respectively, each of size $256 \times 384$ pixels, from the Corel image database. Each nonoverlapping $16 \times 16$ pixels block is called an image site. For each image site $i$, a 5-dim *single-site* feature vector $s_i(y_i)$ and a 14-dim *multiscale* feature vector $f_i(y)$ is computed using orientation and magnitude based features as described in [16]. Note that $f_i(y)$ incorporates data interaction from neighboring sites. For the association potentials, a transformed feature vector $h_i(y)$ was computed at each site $i$ using quadratic transforms of vector $f_i(y)$. The pairwise data vector $\mu_{ij}(y)$ was obtained by concatenating the two vectors $f_i(y)$ and $f_j(y)$. For the DRF parameter learning, $\tau$ was chosen to be $0.001$. For the MRF, each class conditional density was modeled as a mixture of five Gaussians. Use of a single Gaussian for each class yielded very poor results.

For two typical images from the test set, the detection results for the MRF and the DRF models are given in Fig. 2. The blocks detected as *structured* have been shown enclosed within an artificial boundary. The DRF results show higher detections with lower false positives. For a quantitative evaluation, we compared the detection rates and the number of false positives per image for different techniques. For the comparison of detection rates, in all the experiments, the decision threshold of the logistic classifier was fixed such that it yields the same false positive rate as the DRF. The first set of experiments was conducted using the *single-site* features for all the three methods. Thus, no neighborhood data interaction was used for both the logistic classifier and the DRF, i.e. $f_i(y) = s_i(y_i)$. The comparative results for the three methods are given in Table 2 under 'MRF', 'Logistic$^-$' and 'DRF$^-$'. The detection rates of the MRF and the DRF are higher than the logistic classifier due to the label interaction. However, higher detection rate and lower false positives for the DRF in comparison to the MRF indicate the gains due to the use of discriminative models in the association and interaction potentials in the DRF. In the next experiment, to take advantage of the power of the DRF framework, data interaction was allowed for both the logistic classifier as well as the DRF ('Logistic' and 'DRF' in Table 2). The DRF detection rate increases substantially and the false positives decrease further indicating the importance of allowing the data interaction in addition to the label interaction.

## 5 Conclusion and future work

We have presented discriminative random fields which provide a principled approach for combining local discriminative classifiers that allow the use of arbitrary overlapping features, with adaptive data-dependent smoothing over the label field. We are currently exploring alternative ways of parameter learning using contrastive divergence and saddle point approximations. One of the further aspects of the DRF model is the use of general kernel mappings to increase the classification accuracy. However, one will need some method to induce sparseness to avoid overfitting [12]. In addition, we intend to extend the model to accommodate multiclass classification problems.

**Acknowledgments**

Our thanks to John Lafferty and Jonas August for immensely helpful discussions.

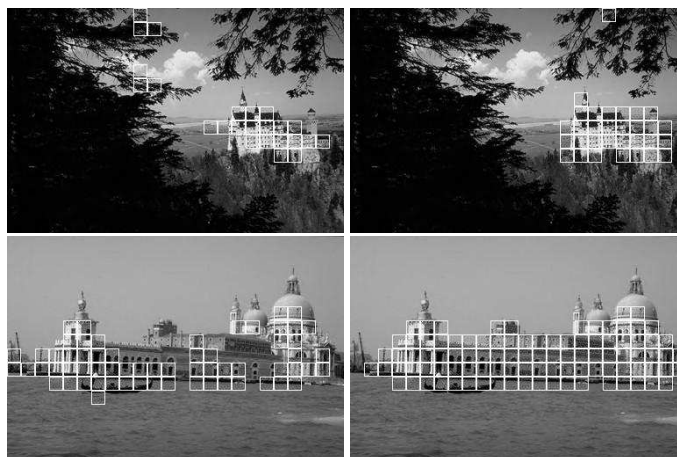

Figure 2: Example structure detection results. Left column: MRF results. Right column: DRF results. DRF has higher detection rate with lower false positives.

## References

[1] S. Z. Li. *Markov Random Field Modeling in Image Analysis*. Springer-Verlag, Tokyo, 2001.

[2] X. Feng, C. K. I. Williams, and S. N. Felderhof. Combining belief networks and neural networks for scene segmentation. *IEEE Trans. Pattern Anal. Machine Intelli.*, 24(4):467–483, 2002.

[3] H. Cheng and C. A. Bouman. Multiscale bayesian segmentation using a trainable context model. *IEEE Trans. on Image Processing*, 10(4):511–525, 2001.

[4] R. Wilson and C. T. Li. A class of discrete multiresolution random fields and its application to image segmentation. *IEEE Trans. on Pattern Anal. and Machine Intelli.*, 25(1):42–56, 2003.

[5] Y. D. Rubinstein and T. Hastie. Discriminative vs informative learning. *In Proc. Third Int. Conf. on Knowledge Discovery and Data Mining*, pages 49–53, 1997.

[6] J. Lafferty, A. McCallum, and F. Pereira. Conditional random fields: Probabilistic models for segmenting and labeling sequence data. *In Proc. Int. Conf. on Machine Learning*, 2001.

[7] S. Kumar and M. Hebert. Discriminative random fields: A discriminative framework for contextual interaction in classification. *IEEE Int. Conf. on Computer Vision*, 2:1150–1157, 2003.

[8] P. McCullagh and J. A. Nelder. *Generalised Linear Models*. Chapman and Hall, London, 1987.

[9] D. M. Greig, B. T. Porteous, and A. H. Seheult. Exact maximum a posteriori estimation for binary images. *Journal of Royal Statis. Soc.*, 51(2):271–279, 1989.

[10] D. Mackay. Bayesian non-linear modelling for the 1993 energy prediction competition. *In Maximum Entropy and Bayesian Methods*, pages 221–234, 1996.

[11] P. Williams. Bayesian regularization and pruning using a laplacian prior. *Neural Computation*, 7:117–143, 1995.

[12] M. A. T. Figueiredo. Adaptive sparseness using jeffreys prior. *Advances in Neural Information Processing Systems (NIPS)*, 2001.

[13] G. E. Hinton. Training product of experts by minimizing contrastive divergence. *Neural Computation*, 14:1771–1800, 2002.

[14] M. Collins. Discriminative training methods for hidden markov models: Theory and experiments with perceptron algorithms. *In Proc. Conference on Empirical Methods in Natural Language Processing (EMNLP)*, 2002.

[15] V. Kolmogorov and R. Zabih. What energy functions can be minimized via graph cuts. *In Proc. European Conf. on Computer Vision*, 3:65–81, 2002.

[16] S. Kumar and M. Hebert. Man-made structure detection in natural images using a causal multiscale random field. *In Proc. IEEE Int. Conf. on Comp. Vision and Pattern Recog.*, June 2003.
